# Fast Prediction on a Tree

**Mark Herbster,   Massimiliano Pontil,   Sergio Rojas-Galeano**
Department of Computer Science
University College London
Gower Street, London WC1E 6BT, England, UK
{*m.herbster, m.pontil,s.rojas*}*@cs.ucl.ac.uk*

## Abstract

Given an $n$-vertex weighted tree with structural diameter $S$ and a subset of $m$ vertices, we present a technique to compute a corresponding $m \times m$ Gram matrix of the pseudoinverse of the graph Laplacian in $O(n + m^2 + mS)$ time. We discuss the application of this technique to fast label prediction on a generic graph. We approximate the graph with a spanning tree and then we predict with the kernel perceptron. We address the approximation of the graph with either a minimum spanning tree or a shortest path tree. The fast computation of the pseudoinverse enables us to address prediction problems on large graphs. We present experiments on two web-spam classification tasks, one of which includes a graph with 400,000 vertices and more than 10,000,000 edges. The results indicate that the accuracy of our technique is competitive with previous methods using the full graph information.

## 1   Introduction

Classification methods which rely upon the graph Laplacian (see [3, 20, 13] and references therein), have proven to be useful for semi-supervised learning. A key insight of these methods is that unlabeled data can be used to improve the performance of supervised learners. These methods reduce to the problem of labeling a graph whose vertices are associated to the data points and the edges to the similarity between pairs of data points. The labeling of the graph can be achieved either in a batch [3, 20] or in an online manner [13]. These methods can all be interpreted as different kernel methods: ridge regression in the case of [3], minimal semi-norm interpolation in [20] or the perceptron algorithm in [13]. This computation scales in the worst case cubically with the quantity of unlabeled data, which may prevent the use of these methods on large graphs.

In this paper, we propose a method to improve the computational complexity of Laplacian-based learning algorithms. If an $n$-vertex tree is given, our method requires an $O(n)$ initialization step and after that any $m \times m$ block of the pseudoinverse of the Laplacian may be computed in $O(m^2 + mS)$ time, where $S$ is the structural diameter of the tree. The pseudoinverse of the Laplacian may then be used as a kernel for a variety of label prediction methods. If a generic graph is given, we first approximate it with a tree and then run our method on the tree. The use of a minimum spanning tree and shortest path tree is discussed.

It is important to note that prediction is also possible using directly the graph Laplacian, without computing its pseudoinverse. For example, this may be achieved by solving a linear system of equations [3, 20] involving the Laplacian, and a solution may be computed in $O(|E| \log^{O(1)} n)$ time [18], where $E$ is the edge set. However, computation via the graph kernel allows for multiple prediction problems on the same graph to be computed more efficiently. The advantage is even more striking if the data come sequentially and we need to predict in an online fashion.

To illustrate the advantage of our approach consider the case in which we are provided with a small subset of $\ell$ labeled vertices of a large graph and we wish to predict the label of a different subset of $p$ vertices. Let $m = \ell + p$ and assume that $m \ll n$ (typically we will also have $\ell \ll p$). A practical application is the problem of detecting "spam" hosts in the internet. Although the number of hosts

in the internet is in the millions we may only need to detect spam hosts from some limited domain. If the graph is a tree the total time required to predict with the kernel perceptron using our method will be $O(n + m^2 + mS)$. The promise of our technique is that, if $m + S \ll n$ and a tree is given, it requires $O(n)$ time versus $O(n^3)$ for standard methods.

To the best of our knowledge this is the first paper which addresses the problem of fast prediction in semi-supervised learning using tree graphs. Previous work has focused on special prediction methods and graphs. The work in [5] presents a non-Laplacian-based method for predicting the labeling of a tree, based on computing the exact probabilities of a Markov random field. The issue of computation time is not addressed there. In the case of unbalanced bipartite graphs [15] presents a method which significantly improves the computation time of the pseudoinverse to $\Theta(k^2(n - k))$, where $k$ is the size of a minority partition. Thus, in the case of a binary tree the computation is still $\Theta(n^3)$ time.

The paper is organized as follows. In Section 2 we review the notions which are needed in order to present our technique in Section 3, concerning the fast computation of a tree graph kernel. In Section 4 we address the issue of tree selection, commenting in particular on a potential advantage of shortest path trees. In Section 5 we present the experimental results and draw our conclusions in Section 6.

## 2 Background

In this paper any graph $\mathcal{G}$ is assumed to be connected, to have $n$ vertices, and to have edge weights. The set of vertices of $\mathcal{G}$ is denoted $V = \{1, \ldots, n\}$. Let $\mathbf{A} = (A_{ij})_{i,j=1}^n$ be the $n \times n$ symmetric weight matrix of the graph, where $A_{ij} \geq 0$, and define the edge set $E(\mathcal{G}) := \{(i, j) : A_{ij} > 0, i < j\}$. We say that $\mathcal{G}$ is a *tree* if it is connected and has $n - 1$ edges. The graph Laplacian is the $n \times n$ matrix defined as $\mathbf{G} = \mathbf{D} - \mathbf{A}$, where $\mathbf{D}$ is the diagonal matrix with $i$-th diagonal element $D_{ii} = \sum_{j=1}^n A_{ij}$, the weighted degree of vertex $i$. Where it is not ambiguous, we will use the notation $\mathbf{G}$ to denote both the graph $\mathcal{G}$ and the graph Laplacian and the notation $\mathbf{T}$ to denote both a Laplacian of a tree and the tree itself. The Laplacian is positive semi-definite and induces the semi-norm $\|\mathbf{w}\|_{\mathbf{G}}^2 := \mathbf{w}^\top \mathbf{G} \mathbf{w} = \sum_{(i,j) \in E(\mathbf{G})} A_{ij}(w_i - w_j)^2$. The kernel associated with the above semi-norm is $\mathbf{G}^+$, the pseudoinverse of matrix $\mathbf{G}$, see e.g. [14] for a discussion. As the graph is connected, it follows from the definition of the semi-norm that the null space of $\mathbf{G}$ is spanned by the constant vector $\mathbf{1}$ only.

The analogy between graphs and networks of resistors plays an important role in this paper. That is, the weighted graph may be seen as a network of resistors where edge $(i, j)$ is a resistor with resistance $\pi_{ij} = A_{ij}^{-1}$. Then the *effective resistance* $r_{\mathbf{G}}(i, j)$ may be defined as the resistance measured between vertex $i$ and $j$ in this network and may be calculated using Kirchoff's circuit laws or directly from $\mathbf{G}^+$ using the formula [16]

$$r_{\mathbf{G}}(i, j) = G_{ii}^+ + G_{jj}^+ - 2G_{ij}^+ . \tag{2.1}$$

The effective resistance is a metric distance on the graph [16] as well as the geodesic and structural distances. The structural distance between vertices $i, j \in V$ is defined as $s_{\mathbf{G}}(i, j) := \min\{|P(i, j)| : P(i, j) \in \mathcal{P}\}$ where $\mathcal{P}$ is the set of all paths in $\mathcal{G}$ and $P(i, j)$ is the set of edges in a particular path from $i$ to $j$. Whereas, the geodesic distance is defined as $d_{\mathbf{G}}(i, j) := \min\{\sum_{(p,q) \in P(i,j)} \pi_{pq} : P(i, j) \in \mathcal{P}\}$. The diameter is the maximum distance between any two points on the graph, hence the resistance, structural, and, geodesic diameter are defined as $R_{\mathbf{G}} = \max_{i,j \in V} r_{\mathbf{G}}(i, j)$ $S_{\mathbf{G}} = \max_{i,j \in V} s_{\mathbf{G}}(i, j)$, and $D_{\mathbf{G}} = \max_{i,j \in V} d_{\mathbf{G}}(i, j)$, respectively. Note that, by Kirchoff's laws, $r_{\mathbf{G}}(i, j) \leq d_{\mathbf{G}}(i, j)$ and, so, $R_{\mathbf{G}} \leq D_{\mathbf{G}}$.

## 3 Computing the Pseudoinverse of a Tree Laplacian Quickly

In this section we describe our method to compute the pseudoinverse of a tree.

### 3.1 Inverse Connectivity

Let us begin by noting that the effective resistance is a better measure of connectivity than the geodesic distance, as for example if there are $k$ edge disjoint paths of geodesic distance $d$ between two vertices, then the effective resistance is no more than $\frac{d}{k}$. Thus, the more paths, the closer the vertices.

In the following, we will introduce three more global measures of connectivity built on top of the effective resistance, which are useful for our computation below. The first quantity is the *total resistance* $R_{\text{tot}} = \sum_{i>j} r_{\mathbf{G}}(i,j)$, which is a measure of the *inverse connectivity* of the graph: the smaller $R_{\text{tot}}$ the more connected the graph. The second quantity is $R(i) = \sum_{j=1}^{n} r_{\mathbf{G}}(i,j)$, which is used as a measure of *inverse centrality* of vertex $i$ [6, Def. 3] (see also [17]). The third quantity is $G_{ii}^{+}$, which provides an alternate notion of inverse centrality.

Summing both sides of equation (2.1) over $j$ gives

$$R(i) = nG_{ii}^{+} + \sum_{j=1}^{n} G_{jj}^{+}, \tag{3.1}$$

where we used the fact that $\sum_{j=1}^{n} G_{ij}^{+} = 0$, which is true because the null space of $\mathbf{G}$ is spanned by the constant vector. Summing again over $i$ yields

$$R_{\text{tot}} = n \sum_{i=1}^{n} G_{ii}^{+}, \tag{3.2}$$

where we have used $\sum_{i=1}^{n} R(i) = 2R_{\text{tot}}$. Combing the last two equations we obtain

$$G_{ii}^{+} = \frac{R(i)}{n} - \frac{R_{\text{tot}}}{n^2}. \tag{3.3}$$

## 3.2 Method

Throughout this section we assume that $\mathcal{G}$ is a tree with corresponding Laplacian matrix $\mathbf{T}$. The principle of the method to compute $\mathbf{T}^{+}$ is that, on a tree there is a *unique* path between any two vertices and, so, the effective resistance is simply the sum of resistances along that path, see e.g. [16, 13] (for the same reason, on a tree the geodesic distance is the same as the resistance distance).

We assume that the root vertex is indexed as $1$. The parent and the children of vertex $i$ are denoted by $\uparrow(i)$ and $\downarrow(i)$, respectively. The descendants of vertex $i$ are denoted by

$$\downarrow^{*}(i) := \begin{cases} \downarrow(i) \bigcup_{j \in \downarrow(i)} \downarrow^{*}(j) & \downarrow(i) \neq \emptyset \\ \emptyset & \downarrow(i) = \emptyset \end{cases}.$$

We also let $\kappa(i)$ be the number of descendants of vertex $i$ and $i$ itself, that is, $\kappa(i) = 1 + |\downarrow^{*}(i)|$.

The method is outlined as follows. We initially compute $R(1), \ldots, R(n)$ in $O(n)$ time. This in turn gives us $R_{\text{tot}} = \frac{1}{2} \sum_{i=1}^{n} R(i)$ and $G_{11}^{+}, \ldots, G_{nn}^{+}$ via equation (3.3), also in $O(n)$ time. As we shall see, with these precomputed values, we may obtain off-diagonal elements $G_{ij}^{+}$ from equation (2.1) by computing individually $r_{\mathbf{T}}(i,j)$ in $O(S_{\mathbf{T}})$ or an $m \times m$ block in $O(m^2 + mS_{\mathbf{T}})$ time.

### Initialization

We split the computation of the inverse centrality $R(i)$ into two terms, namely $R(i) = T(i) + S(i)$, where $T(i)$ and $S(i)$ are the sum of the resistances of vertex $i$ to each descendant and non-descendant, respectively. That is,

$$T(i) = \sum_{j \in \downarrow^{*}(i)} r_{\mathbf{T}}(i,j), \quad S(i) = \sum_{j \notin \downarrow^{*}(i)} r_{\mathbf{T}}(i,j).$$

We compute $\kappa(i)$ and $T(i)$, $i = 1, \ldots, n$ with the following leaves-to-root recursions

$$\kappa(i) := \begin{cases} 1 + \sum_{j \in \downarrow(i)} \kappa(j) & \downarrow(i) \neq \emptyset \\ 1 & \downarrow(i) = \emptyset \end{cases}, \quad T(i) := \begin{cases} \sum_{j \in \downarrow(i)} (T(j) + \pi_{ij}\kappa(j)) & \downarrow(i) \neq \emptyset \\ 0 & \downarrow(i) = \emptyset \end{cases}$$

by computing $\kappa(1)$ then $T(1)$ and caching the intermediate values. We next descend the tree caching each calculated $S(i)$ with the root-to-leaves recursion

$$S(i) := \begin{cases} S(\uparrow(i)) + T(\uparrow(i)) - T(i) + (n - 2\kappa(i))\pi_{i\uparrow(i)} & i \neq 1 \\ 0 & i = 1 \end{cases}.$$

It is clear that the time complexity of the above recursions is $O(n)$.

```
1.   Input: {v_1, ..., v_m} ⊆ V
2.   Initialization: visited(all) = ∅
3.   for i = 1, ..., m do
4.        p = -1; c = v_i; r_T(c, c) = 0
5.      Repeat
6.        for w ∈ visited(c) ∩ {p} ∪ ↓*(p) do
7.           r_T(v_i, w) = r_T(w, v_i) = r_T(v_i, c) + r_T(c, w)
8.        end
9.        visited(c) = visited(c) ∪ v_i
10.       p = c; c = ↑(c)
11.       r_T(v_i, c) = r_T(c, v_i) = r_T(v_i, p) + π_{p,c}
12.     until ("p is the root")
13.  end
```

Figure 1: Computing an $m \times m$ block of a tree Laplacian pseudoinverse.

**Computing an $m \times m$ block of the Laplacian pseudoinverse**

Our algorithm (see Figure 1) computes the effective resistance matrix of an $m \times m$ block which effectively gives the kernel (via equation (2.1)). The motivating idea is that a single effective resistance $r_T(i, j)$ is simply the sum of resistances along the path from $i$ to $j$. It may be computed by separately ascending the path from $i$–to–root and $j$–to–root in $O(S_T)$ time and summing the resistances along each edge that is either in the $i$–to–root or $j$–to–root path but not in both. However we may amortize the computation of an $m \times m$ block to $O(m^2 + mS_T)$ time, saving a factor of $\min(m, S_T)$. This is realized by additionally caching the cumulative sums of resistances along the path to the root during each ascent from a vertex.

We outline in further detail the algorithm as follows: for each vertex $v_i$ in the set $V_m = \{v_1, ..., v_m\}$ we perform an ascent to the root (see line 3 in Figure 1). As we ascend, we cache each cumulative resistance (from the starting vertex $v_i$ to the current vertex $c$) along the path on the way to the root (line 11). If, while ascending from $v_i$ we enter a vertex $c$ which has previously been visited during the ascent from another vertex $w$ (line 6) then we compute $r_T(v_i, w)$ as $r_T(v_i, c) + r_T(c, w)$. For example, during the ascent from vertex $v_k \in V_m$ to the root we will compute $\{r_T(v_1, v_k), ..., r_T(v_k, v_k)\}$.

The computational complexity is obtained by noting that every ascent to the root requires $O(S_T)$ steps and along each ascent we must compute up to $\max(m, S_T)$ resistances. Thus, the total complexity is $O(m^2 + mS_T)$, assuming that each step of the algorithm is efficiently implemented. For this purpose, we give two implementation notes. First, each of the effective resistances computed by the algorithm should be stored on the tree, preventing creation of an $n \times n$ matrix. When the computation is completed the desired $m \times m$ Gram matrix may then be directly computed by gathering the cached values via an additional set of ascents. Second, it should be ensured that the "for loop" (line 6) is executed in $\Theta(|\text{visited}(c) \cap \overline{\{p\} \cup \downarrow^*(p)}|)$ time by a careful but straightforward implementation of the visited predicate. Finally, this algorithm may be generalized to compute a $p \times \ell$ block in $O(p\ell + (p + \ell)S_T)$ time or to operate fully "online."

Let us return to the practical scenario described in the introduction, in which we wish to predict $p$ vertices of the tree based on $\ell$ labeled vertices. Let $m = \ell + p$. By the above discussion, computation of an $m \times m$ block of the kernel matrix $\mathbf{T}^+$ requires $O(n + m^2 + mS_T)$ time. In many practical applications $m \ll n$ and $S_G$ will typically be no more than logarithmic in $n$, which leads to an appealing $O(n)$ time complexity.

## 4   Tree Construction

In the previous discussion, we have considered that a tree has already been given. In the following, we assume that a graph $\mathbf{G}$ or a similarity function is given and the aim is to construct an approximating tree. We will consider both the *minimum spanning tree* (MST) as a "best" in norm approximation; and the *shortest path tree* (SPT) as an approximation which maintains a mistake bound [13] guarantee.

Given a graph with a "cost" on each edge, an MST is a connected $n$-vertex subgraph with $n - 1$ edges such that the total cost is minimized. In our set-up the cost of edge $(i, j)$ is the resistance

$\pi_{ij} = \frac{1}{A_{ij}}$, therefore, a minimum spanning tree of $\mathbf{G}$ solves the problem

$$\min \left\{ \sum_{(i,j) \in E(\mathbf{T})} \pi_{ij} : \mathbf{T} \in \mathcal{T}(\mathbf{G}) \right\}, \tag{4.1}$$

where $\mathcal{T}(\mathbf{G})$ denotes the set of spanning trees of $\mathbf{G}$. An MST is also a tree whose Laplacian best approximates the Laplacian of the given graph according to the trace norm, that is, it solves the problem

$$\min \left\{ \operatorname{tr}(\mathbf{G} - \mathbf{T}) : \mathbf{T} \in \mathcal{T}(\mathbf{G}) \right\}. \tag{4.2}$$

Indeed, we have $\operatorname{tr}(\mathbf{G} - \mathbf{T}) = \sum_{i,j=1}^{n} A_{ij} - \sum_{(i,j) \in E(\mathbf{T})} -\pi_{ij}^{-1}$. Then, our claim that the problems (4.1) and (4.2) have the same solution follows by noting that the edges in a minimum spanning tree are invariant with respect to any strictly increasing function of the "costs" on the edges in the original graph [8] and the function $-\pi^{-1}$ is increasing in $\pi$.

The above observation suggests another approximation criterion which we may consider for finding a spanning tree. We may use the trace norm between the pseudoinverse of the Laplacians, rather than the Laplacians themselves as in (4.2). This seems a more natural criterion, since our goal is to approximate well the kernel (it is the kernel which is directly involved in the prediction problem). It is interesting to note that the quantity $\operatorname{tr}(\mathbf{T}^+ - \mathbf{G}^+)$ is related to the total resistance. Specifically, we have by equation (3.2) that $\operatorname{tr}(\mathbf{T}^+ - \mathbf{G}^+) = \frac{R_{\text{tot}}(\mathbf{T})}{n} - \frac{R_{\text{tot}}(\mathbf{G})}{n}$. As noted in [10], the total resistance is a convex function of the graph Laplacian. However, we do not know how to minimize $R_{\text{tot}}(\mathbf{T})$ over the set of spanning trees of $\mathbf{G}$. We thus take a different route, which leads us to the notion of *shortest path trees*. We choose a vertex $i$ and look for a spanning tree which minimizes the inverse centrality $R(i)$ of vertex $i$, that is we solve the problem

$$\min \left\{ R(i) : \mathbf{T} \in \mathcal{T}(\mathbf{G}) \right\}. \tag{4.3}$$

Note that $R(i)$ is the contribution of vertex $i$ to the total resistance of $\mathbf{T}$ and that, by equations (3.1) and (3.2), $R(i) = nT_{ii}^+ + \frac{R_{\text{tot}}}{n}$. The above problem can then be interpreted as minimizing a trade-off between inverse centrality of vertex $i$ and inverse connectivity of the tree. In other words, (4.3) encourages trees which are centered at $i$ and, at the same time have a small diameter. It is interesting to observe that the solution of problem (4.3) is a shortest path tree (SPT) centered at vertex $i$, namely a spanning tree for which the geodesic distance in "costs" is minimized from $i$ to every other vertex in the graph. This is because the geodesic distance is equivalent to the resistance distance on a tree and any SPT of $\mathbf{G}$ is formed from a set of shortest paths connecting the root to any other vertex in $\mathbf{G}$ [8, Ch. 24.1].

Let us observe a fundamental difference between MST and SPT, which provides a justification for approximating the given graph with an SPT. It relies upon the analysis in [13, Theorem 4.2], where the cumulative number of mistakes of the kernel perceptron with the kernel $\mathbf{K} = \mathbf{G}^+ + \mathbf{1}\mathbf{1}^\top$ was upper bounded by $(\|\mathbf{u}\|_{\mathbf{G}}^2 + 1)(R_{\mathbf{G}} + 1)$ for consistent labelings [13] $\mathbf{u} \in \{-1, 1\}^n$. To explain our argument, first we note that when we approximate the graph with a tree $\mathbf{T}$ the term $\|\mathbf{u}\|_{\mathbf{G}}^2$ is always decreasing, while the term $R_{\mathbf{G}}$ is always increasing by Rayleigh's monotonicity law (see for example [13, Corollary 3.1]). Now, note that the resistance diameter $R_{\mathbf{T}}$ of an SPT of a graph $\mathbf{G}$ is bounded by twice the geodesic diameter of the original graph,

$$R_{\mathbf{T}} \leq 2D_{\mathbf{G}}. \tag{4.4}$$

Indeed, as an SPT is formed from a set of shortest paths between the root and any other vertex in $\mathbf{G}$, for any pair of vertices $p, q$ in the graph there is in the SPT a path from $p$ to the root and then to $q$ which can be no longer than $2D_{\mathbf{G}}$.

To further discuss, consider the case that $\mathbf{G}$ consists of a few dense clusters each uniquely labeled and with only a few cross-cluster edges. The above mistake bound and the bound (4.4), imply that a tree built with an SPT would still have a non-vacuous mistake bound. No such bound as (4.4) holds for an MST subgraph. For example, consider a bicycle wheel graph whose edge set is the union of $n$ *spoke* edges $\{(0, i) : i = 1, \ldots, n\}$ and $n$ *rim* edges $\{(i, i+1 \mod n) : i = 1, \ldots, n\}$ with costs on the spoke edges of 2 and on the rim edges of 1. The MST diameter is then $n + 1$ while any SPT diameter is $\leq 8$.

At last, let us comment on the time and space complexity of constructing such trees. The MST and SPT trees may be constructed with Prim and Dijkstra algorithms [8] respectively in $O(n \log n + |E|)$ time. Prim' algorithm may be further speeded up to $O(n + |E|)$ time in the case of small integer weights [12]. In the general case of a non-sparse graph or similarity function the time complexity is $\Theta(n^2)$, however as both Prim and Dijkstra are "greedy" algorithms their space complexity is $O(n)$ which may be a dominant consideration in a large graph.

## 5    Web-spam Detection Experiments

In this section, we present an experimental study of the feasibility of our method on large graphs (400,000 vertices). The motivation for our methodology is that on graphs with already 10,000 vertices it is computationally challenging to use standard graph labeling methods such as [3, 20, 13], as they require the computation of the full graph Laplacian kernel. This computational burden makes the use of such methods prohibitive when the number of vertices is in the millions. On the other hand, in the practical scenario described in the introduction the computational time of our method scales linearly in the number of vertices in the graph and can be run comfortably on large graphs (see Figure 2 below) and at worst quadratically if the full graph needs to be labeled.

The aims of the experiments are: (*i*) to see whether there is a significant performance loss when using a tree sub-graph rather than the original graph, (*ii*) to compare tree construction methods, specifically the MST and the SPT and (*iii*) to exploit the possibility of improving performance through ensembles of trees. The initial results are promising in that the performance of the predictor with a single SPT or MST is competitive with that of the existing methods, some of which use the full graph information. We shall also comment on the computational time of the method.

### 5.1    Datasets and previous methods

We applied the Fast Prediction on a Tree (FPT) method to the 2007 web-spam challenge developed at the University of Paris VI[1]. Two graphs are provided. The first one is formed by 9,072 vertices and 464,959 edges, which represent computer hosts – we call this the *host-graph*. In this graph, one host is "connected" to another host if there is at least one link from a web-page in the first host to a web-page in the other host. The second graph consists of 400,000 vertices (corresponding to web-pages) and 10,455,545 edges – we call this the *web-graph*. Again, a web-page is "connected" to another web-page if there is at least one hyperlink from the former to the latter. Note that both graphs are directed. In our experiments we discarded directional information and assigned a weight of either 1 to unidirectional edges and of $w \in \{1, 2\}$ to the bidirectional edges. Each vertex is either labeled as spam or as non-spam. In both graphs there are about 80% of non-spam vertices and 20% of spam ones. Additional tf-idf feature vectors (determined by the web-pages' html content) are provided for each vertex in the graph, but we have discarded this information for simplicity. Following the web-spam protocol, for both graphs we used 10% of labeled vertices for training and 90% for testing.

We briefly discuss some previous methods which participated in the web-spam challenge. *Abernathy et al.* [1] used an SVM variant on the tf-idf features with an additional graph-based regularization term, which penalizes predictions with links between non-spam to spam vertices. *Tang et al.* (see [7]) used a linear and Gaussian SVM combined with Random Forests on the feature vectors, plus new features obtained from link information. The method of *Witschel and Biemann* [4] consisted of iteratively selecting vertices and classifying them with the predominant class in their neighborhood (hence this is very similar to label propagation method of [20]). *Benczúr et al.* (see [7]) used Naive Bayes, C4.5 and SVM's with a combination of content and/or graph-based features. Finally, *Filoche et al.* (see [7]) applied html preprocessing to obtain web-page fingerprints, which were used to obtain clusters; these clusters along with link and content-based features were then fed to a modified Naive Bayes classifier.

### 5.2    Results

Experimental results are shown in Table 1. We report the following performance measures: (*i*) average accuracy when predicting with a single tree, (*ii*) average accuracy when each predictor is optimized over a threshold in the range of $[-1, 1]$, (*iii*) area under the curve (AUC) and (*iv*)

| Method | Agg. | Agg.-Best | AUC | Single | Single-Best | AUC |
|---|---|---|---|---|---|---|
| *Host-graph* | | | | | | |
| MST | 0.907 | 0.907 | 0.950 | 0.857±0.022 | 0.865±0.017 | 0.841±0.045 |
| SPT | 0.889 | 0.890 | 0.952 | 0.850±0.026 | 0.857±0.018 | 0.804±0.063 |
| MST (bidir) | 0.912 | **0.915** | 0.944 | 0.878±0.033 | 0.887±0.027 | 0.851±0.100 |
| SPT (bidir) | **0.913** | 0.913 | **0.960** | 0.873±0.028 | 0.877±0.026 | 0.846±0.065 |
| Abernathy *et al.* | 0.896 | 0.906 | 0.952 | … | … | … |
| Tang *et al.* | 0.906 | 0.907 | 0.951 | … | … | … |
| Filoche *et al.* | 0.889 | 0.890 | 0.927 | … | … | … |
| Benczúr *et al.* | 0.829 | 0.847 | 0.877 | … | … | … |
| *Web-graph* | | | | | | |
| MST (bidir) | 0.991 | 0.992 | **1.000** | 0.976±0.011 | 0.980±0.009 | 0.993±0.005 |
| SPT (bidir) | 0.994 | 0.994 | 0.999 | 0.985±0.002 | 0.985±0.002 | 0.992±0.003 |
| Witschel *et al.* | **0.995** | **0.996** | 0.998 | … | … | … |
| Filoche *et al.* | 0.973 | 0.974 | 0.991 | … | … | … |
| Benczúr *et al.* | 0.942 | 0.942 | 0.973 | … | … | … |
| Tang *et al.* | 0.296 | 0.965 | 0.989 | … | … | … |

Table 1: Results of our FPT method and other competing methods.

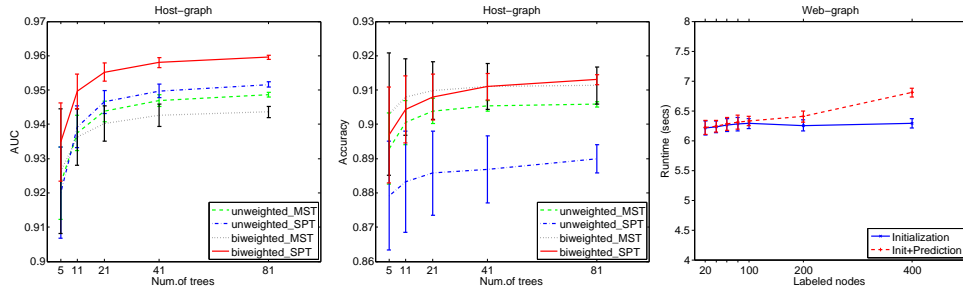

Figure 2: AUC and Accuracy vs. number of trees (left and middle) and Runtime vs. number of labeled vertices (right).

aggregate predictive value given by each tree. In the case of the host-graph, predictions for the aggregate method were made using 81 trees. MST and SPT were obtained for the weighted graphs with Prim and Dijkstra algorithms, respectively. For the unweighted graphs, every tree is an MST, so we simply used trees generated by a randomized unweighted depth-first traversal of the graph and SPT's may be generated by using the breadth-first-search algorithm, all in $O(|E|)$ time. In the table, the tag "Agg." stands for aggregate and the "bidir" tag indicates that the original graph was modified by setting $w = 2$ for bidirectional edges. In the case of the larger web-graph, we used 21 trees and the modified graph with bidirectional weights. In all experiments we used a kernel perceptron which was trained for three epochs (e.g. [13]).

It is interesting to note that some of the previous methods [1, 4] take the full graph information into account. Thus, the above results indicate that our method is statistically competitive (in fact better than most of the other methods) even though the full graph structure is discarded. Remarkably, in the case of the large web-graph, using just a single tree gives a very good accuracy, particularly in the case of SPT. On this graph SPT is also more stable in terms of variance than MST. In the case of the smaller host-graph, just using one tree leads to a decrease in performance. However, by aggregating a few trees our result improves over the state of the art results.

In order to better understand the role of the number of trees on the aggregate prediction, we also ran additional experiments on the host-graph with $t = 5, 11, 21, 41, 81$ randomly chosen MST or SPT trees. We averaged the accuracy and AUC over 100 trials each. Results are shown in Figure 2. As it can be seen, using as few as 11 trees already gives competitive performance. SPT works better than MST in term of AUC (left plot), whereas the result is less clear in the case of accuracy (middle plot).

Finally, we report on an experiment evaluating the running time of our method. We choose the web-graph ($n = 400,000$). We then fixed $p = 1000$ predictive vertices and let the number of labeled vertices $\ell$ vary in the set $\{20, 40, 60, 80, 100, 200, 400\}$. Initialization time (tree construction plus computation of the diagonal elements of the kernel) and initialization plus prediction times were measured in seconds on a dual core 1.8GHz machine with 8Gb memory. As expected, the solid curve, corresponding to initialization time, is the dominant contribution to the computation time.

## 6 Conclusions

We have presented a fast method for labeling of a tree. The method is simple to implement and, in the practical regime of small labeled and testing sets and diameters, scales linearly in the number of vertices in the tree. When we are presented with a generic undirected weighted graph, we first extract a spanning tree from it and then run the method. We have studied minimum spanning trees and shortest path trees, both of which can be computed efficiently with standard algorithms. We have tested the method on a web-spam classification problem involving a graph of 400,000 vertices. Our results indicate that the method is competitive with the state of the art. We have also shown how performance may be improved by averaging the predictors obtained by a few spanning trees. Further improvement may involve learning combinations of different trees. This may be obtained following ideas in [2]. At the same time it would be valuble to study connections between our work and other approximation methods such as those in in the context of kernel-methods [9], Gaussian processes [19] and Bayesian learning [11].

**Acknowledgments.** We wish to thank A. Argyriou and J.-L. Balcázar for valuable discussions, D. Athanasakis and S. Shankar Raman for useful preliminary experimentation, D. Fernandez-Reyes for both useful discussions and computing facility support, and the anonymous reviewers for useful comments. This work was supported in part by the IST Programme of the European Community, under the PASCAL Network of Excellence, IST-2002-506778, by EPSRC Grant EP/D071542/1 and by the DHPA Research Councils UK Scheme.

## Footnotes

[1]See *http://webspam.lip6.fr/wiki/pmwiki.php* for more information.

## References

[1] J. Abernethy, O. Chapelle and C. Castillo. Webspam Identification Through Content and Hyperlinks. Proc. Adversarial Information Retrieval on Web, 2008.

[2] A. Argyriou, M. Herbster, and M. Pontil. Combining graph Laplacians for semi-supervised learning. Advances in Neural Information Processing Systems 17. MIT Press, Cambridge, MA, 2005.

[3] M. Belkin, I. Matveeva, P. Niyogi. Regularization and Semi-supervised Learning on Large Graphs. Proceedings of the 17-th Conference on Learning Theory (COLT' 04), pages 624–638, 2004.

[4] C. Biemann. Chinese Whispers – an Efficient Graph Clustering Algorithm and its Application to Natural Language Processing Problems. Proc. HLT-NAACL-06 Workshop on Textgraphs-06, 2006.

[5] A. Blum, J. Lafferty, M. R. Rwebangira, and R. Reddy. Semi-supervised learning using randomized mincuts. Proc. 21-st International Conference on Machine Learning, page 13, 2004.

[6] U. Brandes and D. Fleischer. Centrality measures based on current flow. Proc. 22-nd Annual Symposium on Theoretical Aspects of Computer Science, pages 533–544, 2005.

[7] C. Castillo, B. D. Davison, L. Denoyer and P. Gallinari. Proc. of the Graph Labelling Workshop and Web-spam Challenge (ECML Workshop), 2007.

[8] T. H. Cormen, C. E. Leiserson, and R. L. Rivest. *Introduction to Algorithms*. MIT Press, 1990.

[9] P. Drineas and M. W. Mahoney, On the Nyström Method for Approximating a Gram Matrix for Improved Kernel-Based Learning. *J. Mach. Learn. Res.*, 6:2153–2175, 2005.

[10] A. Ghosh, S. Boyd and A. Saberi. Minimizing Effective Resistance of a Graph. *SIAM Review, problems and techniques section*, 50(1):37-66, 2008.

[11] T. Jebara. Bayesian Out-Trees. Proc. Uncertainty in Artifical Intelligence, 2008.

[12] R. E. Haymond, J. Jarvis and D. R. Shier. Algorithm 613: Minimum Spanning Tree for Moderate Integer Weights. *ACM Trans. Math. Softw.*, 10(1):108–111, 1984.

[13] M. Herbster and M. Pontil. Prediction on a graph with a perceptron. *Advances in Neural Information Processing Systems 19*, pages 577–584. MIT Press, 2007.

[14] M. Herbster, M. Pontil, and L. Wainer. Online learning over graphs. In *ICML '05: Proceedings of the 22nd international conference on Machine learning*, pages 305–312, 2005.

[15] N.-D. Ho and P. V. Dooren. On the pseudo-inverse of the Laplacian of a bipartite graph. *Appl. Math. Lett.*, 18(8):917–922, 2005.

[16] D. Klein and M. Randić. Resistance distance. *J. of Mathematical Chemistry*, 12(1):81–95, 1993.

[17] M. E. J. Newman. A measure of betweenness centrality based on random walks. *Soc. Networks*, 27:39–54, 2005.

[18] D. A. Spielman and S.-H. Teng. Nearly-linear time algorithms for graph partitioning, graph sparsification, and solving linear systems. Proc. 36-th Annual ACM Symposium Theory of Computing, 2004.

[19] C.K.I. Williams and M. Seeger. Using the Nyström Method to Speed Up Kernel Machines. Neural Information Processing Systems 13, pages 682–688, MIT Press, 2001

[20] X. Zhu, J. Lafferty, and Z. Ghahramani. Semi-Supervised Learning Using Gaussian Fields and Harmonic Functions. Proc of the the 20-th International Conference on Machine Learning, pages 912–919, 2003.